# Spectral Clustering with Perturbed Data

**Ling Huang**
Intel Research
ling.huang@intel.com

**Donghui Yan**
UC Berkeley
dhyan@stat.berkeley.edu

**Michael I. Jordan**
UC Berkeley
jordan@cs.berkeley.edu

**Nina Taft**
Intel Research
nina.taft@intel.com

## Abstract

Spectral clustering is useful for a wide-ranging set of applications in areas such as biological data analysis, image processing and data mining. However, the computational and/or communication resources required by the method in processing large-scale data are often prohibitively high, and practitioners are often required to perturb the original data in various ways (quantization, downsampling, etc) before invoking a spectral algorithm. In this paper, we use stochastic perturbation theory to study the effects of data perturbation on the performance of spectral clustering. We show that the error under perturbation of spectral clustering is closely related to the perturbation of the eigenvectors of the Laplacian matrix. From this result we derive approximate upper bounds on the clustering error. We show that this bound is tight empirically across a wide range of problems, suggesting that it can be used in practical settings to determine the amount of data reduction allowed in order to meet a specification of permitted loss in clustering performance.

## 1 Introduction

A critical problem in machine learning is that of scaling: Algorithms should be effective computationally and statistically as various dimensions of a problem are scaled. One general tool for approaching large-scale problems is that of clustering or partitioning, in essence an appeal to the principle of divide-and-conquer. However, while the output of a clustering algorithm may yield a set of smaller-scale problems that may be easier to tackle, clustering algorithms can themselves be complex, and large-scale clustering often requires the kinds of preprocessing steps that are invoked for other machine learning algorithms [1], including proto-clustering steps such as quantization, downsampling and compression. Such preprocessing steps also arise in the distributed sensing and distributed computing setting, where communication and storage limitations may preclude transmitting the original data to centralized processors.

A number of recent works have begun to tackle the issue of determining the tradeoffs that arise under various "perturbations" of data, including quantization and downsampling [2, 3, 4]. Most of these analyses have been undertaken in the context of well-studied domains such as classification, regression and density estimation, for which there are existing statistical analyses of the effect of noise on performance. Although extrinsic noise differs conceptually from perturbations to data imposed by a data analyst to cope with resource limitations, the mathematical issues arising in the two cases are similar and the analyses of noise have provided a basis for the study of the tradeoffs arising from perturbations.

In this paper we focus on spectral clustering, a class of clustering methods that are based on eigen-decompositions of affinity, dissimilarity or kernel matrices [5, 6, 7, 8]. These algorithms often outperform traditional clustering algorithms such as the K-means algorithm or hierarchical clustering. To date, however, their impact on real-world, large-scale problems has been limited; in particular, a distributed or "in-network" version of spectral clustering has not yet appeared. Moreover, there has been little work on the statistical analysis of spectral clustering, and thus there is little theory to guide the design of distributed algorithms. There is an existing literature on numerical techniques for

**Procedure** SpectralClustering $(\mathbf{x}_1, \ldots, \mathbf{x}_n)$
**Input**: $n$ data samples $\{\mathbf{x}_i\}_{i=1}^n, \mathbf{x}_i \in \mathbb{R}^d$
**Output**: Bipartition $S$ and $\bar{S}$ of the input data

1. Compute the similarity matrix $K$:
$$K_{ij} = \exp\left(-\frac{\|\mathbf{x}_i - \mathbf{x}_j\|^2}{2\sigma_k^2}\right), \forall \mathbf{x}_i, \mathbf{x}_j$$
2. Compute the diagonal degree matrix D:
$$D_i = \sum_{j=1}^n K_{ij}$$
3. Compute the normalized Laplacian matrix:
$$L = I - D^{-1}K$$
4. Find the second eigenvector $\mathbf{v}_2$ of $L$
5. Obtain the two partitions using $\mathbf{v}_2$:
6. $\quad S = \{[i] : v_{2i} > 0\}, \quad \bar{S} = \{[i] : v_{2i} \leq 0\}$

Figure 1: A spectral bipartitioning algorithm.

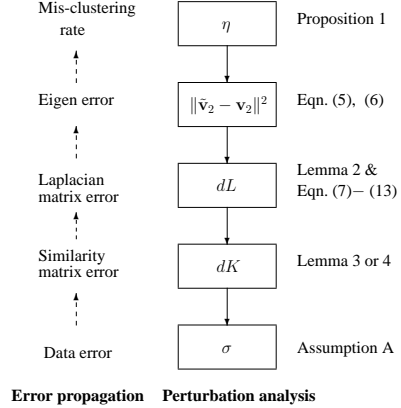

Figure 2: Perturbation analysis: from clustering error to data perturbation error.

scaling spectral clustering (including downsampling [9, 10] and the relaxation of precision requirements for the eigenvector computation [7]), but this literature does not provide end-to-end, practical bounds on error rates as a function of data perturbations.

In this paper we present the first end-to-end analysis of the effect of data perturbations on spectral clustering. Our focus is quantization, but our analysis is general and can be used to treat other kinds of data perturbation. Indeed, given that our approach is based on treating perturbations as random variables, we believe that our methods will also prove useful in developing statistical analyses of spectral clustering (although that is not our focus in this paper).

The paper is organized as follows. In Section 2, we provide a brief introduction to spectral clustering. Section 3 contains the main results of the paper; specifically we introduce the mis-clustering rate $\eta$, and present upper bounds on $\eta$ due to data perturbations. In Section 4, we present an empirical evaluation of our analyses. Finally, in Section 5 we present our conclusions.

## 2 Spectral clustering and data perturbation

### 2.1 Background on spectral clustering algorithms

Given a set of data points $\{\mathbf{x}_i\}_{i=1}^n, \mathbf{x}_i \in \mathbb{R}^{1 \times d}$ and some notion of similarity between all pairs of data points $\mathbf{x}_i$ and $\mathbf{x}_j$, spectral clustering attempts to divide the data points into groups such that points in the same group are similar and points in different groups are dissimilar. The point of departure of a spectral clustering algorithm is a weighted *similarity graph* $G(V, E)$, where the vertices correspond to data points and the weights correspond to the pairwise similarities. Based on this weighted graph, spectral clustering algorithms form the graph Laplacian and compute an eigendecomposition of this Laplacian [5, 6, 7]. While some algorithms use multiple eigenvectors and find a $k$-way clustering directly, the most widely studied algorithms form a bipartitioning of the data by thresholding the second eigenvector of the Laplacian (the eigenvector with the second smallest eigenvalue). Larger numbers of clusters are found by applying the bipartitioning algorithm recursively. We present a specific example of a spectral bipartitioning algorithm in Fig. 1.

### 2.2 Input data perturbation

Let the data matrix $X \in \mathbb{R}^{n \times d}$ be formed by stacking $n$ data samples in rows. To this data matrix we assume that perturbation $W$ is applied, such that we obtain a perturbed version $\tilde{X}$ of the original data $X$. We assume that a spectral clustering algorithm is applied to $\tilde{X}$ and we wish to compare the results of this clustering with respect to the spectral clustering of $X$. This analysis captures a number of data perturbation methods, including data filtering, quantization, lossy compression and synopsis-based data approximation [11]. The multi-scale clustering algorithms that use "representative" samples to approximate the original data can be treated using our analysis as well [12].

# 3 Mis-clustering rate and effects of data perturbation

Let $K$ and $L$ be the similarity and Laplacian matrix on the original data $X$, and let $\tilde{K}$ and $\tilde{L}$ be those on the perturbed data. We define the *mis-clustering rate* $\eta$ as the proportion of samples that have different cluster memberships when computed on the two different versions of the data, $X$ and $\tilde{X}$. We wish to bound $\eta$ in terms of the "magnitude" of the error matrix $W = \tilde{X} - X$, which we now define. We make the following general stochastic assumption on the error matrix $W$:

> A. All elements of the error matrix $W$ are i.i.d. random variables with zero mean, bounded variance $\sigma^2$ and bounded fourth central moment $\mu^4$; and are independent of $X$.

**Remark.** (i) Note that we do not make i.i.d. assumptions on the elements of the similarity matrix; rather, our assumption refers to the input data only. (ii) This assumption is distribution free, and captures a wide variety of practical data collection and quantization schemes. (iii) Certain data perturbation schemes may not satisfy the independence assumption. We have not yet conducted an analysis of the robustness of our bounds to lack of independence, but in our empirical work we have found that the bounds are robust to relatively small amounts of correlation.

We aim to produce practically useful bounds on $\eta$ in terms of $\sigma$ and the data matrix $X$. The bounds should be reasonably tight so that in practice they could be used to determine the degree of perturbation $\sigma$ given a desired level of clustering performance, or to provide a clustering error guarantee on the original data even though we have access only to its approximate version.

Fig. 2 outlines the steps in our theoretical analysis. Briefly, when we perturb the input data (e.g., by filtering, quantization or compression), we introduce a perturbation $W$ to the data which is quantified by $\sigma^2$. This induces an error $dK := \tilde{K} - K$ in the similarity matrix, and in turn an error $dL := \tilde{L} - L$ in the Laplacian matrix. This further yields an error in the second eigenvector of the Laplacian matrix, which results in mis-clustering error. Overall, we establish an analytical relationship between the mis-clustering rate $\eta$ and the data perturbation error $\sigma^2$, where $\eta$ is usually monotonically increasing with $\sigma^2$. Our goal is to allow practitioners to specify a mis-clustering rate $\eta^*$, and by inverting this relationship, to determine the right magnitude of the perturbation $\sigma^*$ allowed. That is, our work can provide a practical method to determine the tradeoff between data perturbation and the loss of clustering accuracy due to the use of $\tilde{X}$ instead of $X$. When the data perturbation can be related to computational or communications savings, then our analysis yields a practical characterization of the overall resource/accuracy tradeoff.

**Practical Applications**  Consider in particular a clustering task in a distributed networking system that allows an application to specify a desired clustering error $C^*$ on the distributed data (which is not available to the coordinator). Through a communication protocol similar to that in [4], the coordinator (e.g., network operation center) gets access to the perturbed data $\tilde{X}$ for spectral clustering. The coordinator can compute a clustering error bound $C$ using our method. By setting $C \leq C^*$, it determines the tolerable data perturbation error $\sigma^*$ and instructs distributed devices to use appropriate numbers of bits to quantize their data. Thus we can provide guarantees on the achieved error, $C \leq C^*$, with respect to the original distributed data even with access only to the perturbed data.

## 3.1 Upper bounding the mis-clustering rate

Little is currently known about the connection between clustering error and perturbations to the Laplacian matrix in the spectral clustering setting. [5] presented an upper bound for the clustering error, however this bound is usually quite loose and is not viable for practical applications. In this section we propose a new approach based on a water-filling argument that yields a tighter, practical bound. Let $\mathbf{v}_2$ and $\tilde{\mathbf{v}}_2$ be the unit-length second eigenvectors of $L$ and $\tilde{L}$, respectively. We derive a relationship between the mis-clustering rate $\eta$ and $\delta^2 := \|\tilde{\mathbf{v}}_2 - \mathbf{v}_2\|^2$.

The intuition behind our derivation is suggested in Fig. 3. Let $a$ and $b$ denote the sets of components in $\mathbf{v}_2$ corresponding to clusters of size $k_1$ and $k_2$, respectively, and similarly for $a'$ and $b'$ in the case of $\tilde{\mathbf{v}}_2$. If $\mathbf{v}_2$ is changed to $\tilde{\mathbf{v}}_2$ due to the perturbation, an incorrect clustering happens whenever a component of $\mathbf{v}_2$ in set $a$ jumps to set $b'$, denoted as $a \to b'$, or a component in set $b$ jumps to set $a'$, denoted as $b \to a'$. The key observation is that each flipping of cluster membership in either $a \to b'$

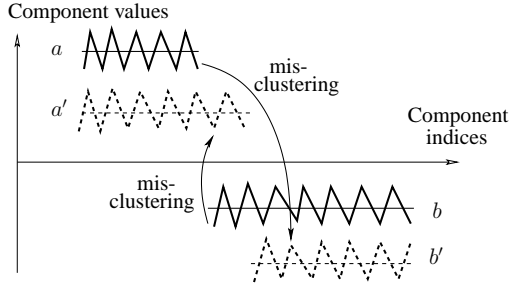

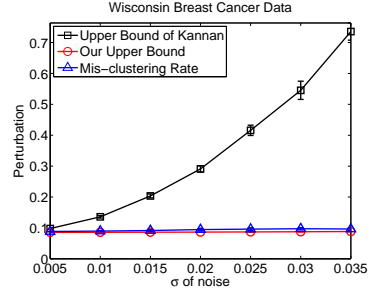

Figure 3: The second eigenvector $\mathbf{v}_2$ and its perturbed counterpart $\tilde{\mathbf{v}}_2$ (denoted by dashed lines).

Figure 4: An example of the tightness of the upper bound for $\eta$ in Eq. (1).

or $b \rightarrow a'$ contributes a fairly large amount to the value of $\delta^2$, compared to the short-range drifts in $a \rightarrow a'$ or $b \rightarrow b'$. Given a fixed value of $\delta^2$, the maximum possible number of flippings (i.e., missed clusterings) is therefore constrained, and this translates into an upper bound for $\eta$.

We make the following assumptions on the data $X$ and its perturbation:

- B1. The components of $\mathbf{v}_2$ form two clusters (with respect to the spectral bipartitioning algorithm in Fig. 1). The size of each cluster is comparable to $n$.
- B2. The perturbation is small with the total number of mis-clusterings $m < \min(k_1, k_2)$, and the components of $\tilde{\mathbf{v}}_2$ form two clusters. The size of each cluster is comparable to $n$.
- B3. The perturbation of individual components of $\mathbf{v}_2$ in each set of $a \rightarrow a'$, $a \rightarrow b'$, $b \rightarrow a'$ and $b \rightarrow b'$ have identical (not necessary independent) distributions with bounded second moments, respectively, and they are uncorrelated with the components in $\mathbf{v}_2$.

Our perturbation bound can now be stated as follows:

**Proposition 1.** *Under assumptions B1, B2 and B3, the mis-clustering rate $\eta$ of the spectral bipartitioning algorithm under the perturbation satisfies $\eta \leq \delta^2 = \|\tilde{\mathbf{v}}_2 - \mathbf{v}_2\|^2$. If we further assume that all components of $\tilde{\mathbf{v}}_2 - \mathbf{v}_2$ are independent, then*

$$\eta \leq (1 + o_p(1))\mathrm{E}\|\tilde{\mathbf{v}}_2 - \mathbf{v}_2\|^2. \tag{1}$$

The proof of the proposition is provided in the Appendix.

**Remarks.** (i) Assumption B3 was motivated by our empirical work. Although it is difficult to establish general necessary and sufficient conditions for B3 to hold, in the Appendix we present some special cases that allow B3 to be verified a priori. It is also worth noting that B3 appears to hold (approximately) across a range of experiments presented in Section 4. (ii) If we assume piecewise constancy for $\mathbf{v}_2$, then we can relax the uncorrelated assumption in B3. (iii) Our bound has a different flavor than that obtained in [5]. Although the bound in Theorem 4.3 in [5] works for $k$-way clustering, it assumes a block-diagonal Laplacian matrix and requires the gap between the $k^{th}$ and $(k + 1)^{th}$ eigenvalues to be greater than $1/2$, which is unrealistic in many data sets. In the setting of 2-way spectral clustering and a small perturbation, our bound is much tighter than that derived in [5]; see Fig. 4 in particular.

## 3.2 Perturbation on the second eigenvector of Laplacian matrix

We now turn to the relationship between the perturbation of eigenvectors with that of its matrix. One approach is to simply draw on the classical domain of matrix perturbation theory; in particular, applying Theorem V.2.8 from [13], we have the following bound on the (small) perturbation of the second eigenvector:

$$\|\tilde{\mathbf{v}}_2 - \mathbf{v}_2\| \leq \frac{\|4dL\|_F}{\nu - \sqrt{2}\|dL\|_F}, \tag{2}$$

where $\nu$ is the gap between the second and the third eigenvalue. However, in our experimental evaluation we found that $\nu$ can be quite small in some data sets, and in these cases the right-hand

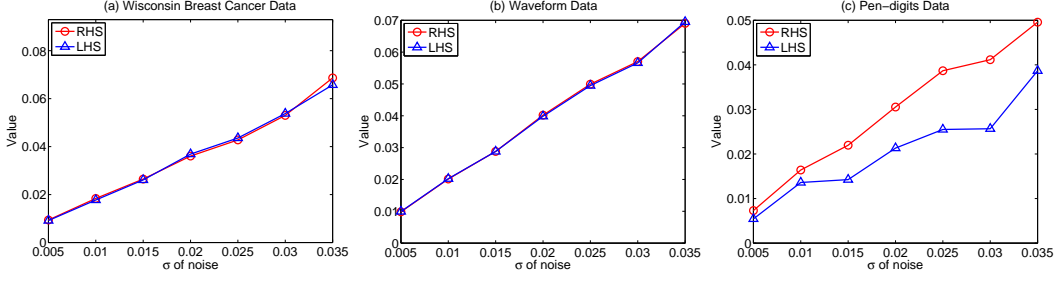

Figure 5: Experimental examples of the fidelity of the approximation in Eq. (5). We add i.i.d. zero mean Gaussian noise to the input data with different $\sigma$, and we see that the right-hand side (RHS) of (5) approximately upper bounds the left-hand side (LHS).

side of (2) can be quite large even for a small perturbation. Thus the bound given by (2) is often not useful in practical applications.

To derive a more practically useful bound, we begin with a well-known first-order Taylor expansion to compute the perturbation on the second eigenvector of a Laplacian matrix as follows:

$$
\begin{aligned}
\tilde{\mathbf{v}}_2 - \mathbf{v}_2 &= \sum_{j=1,j\neq 2}^{n} \frac{\mathbf{v}_j^T dL \mathbf{v}_2}{\lambda_2 - \lambda_j} \mathbf{v}_j + O(dL^2) \approx \sum_{j=1,j\neq 2}^{n} \frac{\mathbf{v}_j}{\lambda_2 - \lambda_j} \sum_{p=1}^{n}\sum_{q=1}^{n} v_{pj} v_{q2} dL_{pq} \\
&= \sum_{p=1}^{n} \left[ \left( \sum_{q=1}^{n} v_{q2} dL_{pq} \right) \cdot \left( \sum_{j=1,j\neq 2}^{n} \frac{v_{pj}\cdot \mathbf{v}_j}{\lambda_2 - \lambda_j} \right) \right] = \sum_{p=1}^{n} \beta_p \mathbf{u}_p,
\end{aligned}
\tag{3}
$$

where $\beta_p = \sum_{q=1}^{n} v_{q2} dL_{pq}$ is a random variable determined by the effect of the perturbation on the Laplacian matrix $L$, and the vector $\mathbf{u}_p = \sum_{j=1,j\neq 2}^{n} \left( \frac{v_{pj}\mathbf{v}_j}{\lambda_2 - \lambda_j} \right)$ is a constant determined by the eigendecomposition of the Laplacian matrix $L$. Then we have

$$
\mathrm{E}\|\tilde{\mathbf{v}}_2 - \mathbf{v}_2\|^2 \approx \mathrm{E}\left\| \sum_{p=1}^{n} \beta_p \mathbf{u}_p \right\|^2 = \sum_{p=1}^{n} \mathrm{E}\|\beta_p \mathbf{u}_p\|^2 + 2\sum_{i=1}^{n}\sum_{j=i+1}^{n} \mathrm{E}\left( \beta_i \mathbf{u}_i \cdot \beta_j \mathbf{u}_j^T \right).
\tag{4}
$$

In our experimental work we have found that for $i \neq j$, $\beta_i \mathbf{u}_i$ is either very weakly correlated with $\beta_j \mathbf{u}_j$ (i.e., the total sum of all cross terms is typically one or two orders of magnitude less than that of squared term), or negatively correlated with $\beta_j \mathbf{u}_j$ (i.e., the total sum of all cross terms is less than zero). This empirical evidence suggests the following approximate bound:

$$
\mathrm{E}\|\tilde{\mathbf{v}}_2 - \mathbf{v}_2\|^2 \lesssim \sum_{p=1}^{n} \mathrm{E}\beta_p^2 \cdot \|\mathbf{u}_p\|^2.
\tag{5}
$$

Examples of the fidelity of this approximation for particular data sets are shown in Fig. 5.

Finally, $\mathrm{E}\beta_p^2$ is related to $dL_{pq}$, and can be upper bounded by

$$
\mathrm{E}\beta_p^2 = \mathrm{E}\left( \sum_{q=1}^{n} v_{q2} dL_{pq} \right)^2 \leq \sum_{i=1}^{n}\sum_{j=1}^{n} \left[ v_{i2} v_{j2} \cdot \mathrm{E}\left( dL_{pi} \right) \mathrm{E}\left( dL_{pj} \right) + |v_{i2} v_{j2}| \sigma_{pi} \sigma_{pj} \right],
\tag{6}
$$

where $\sigma_{pi}$ is the variance of $dL_{pi}$.

**Remark.** Through Eqs. (5) and (6), we can bound the squared norm of the perturbation on the second eigenvector in expectation, which in turn bounds the mis-clustering rate. To compute the bound, we need to estimate the first two moments of $dL$, which we discuss next.

### 3.3 Perturbation on the Laplacian matrix

Let $D$ be the diagonal matrix with $D_i = \sum_j K_{ij}$. We define the normalized Laplacian matrix as $L = I - D^{-1}K$. Letting $\Delta = \tilde{D} - D$ and $dK = \tilde{K} - K$, we have the following approximation for $dL = \tilde{L} - L$:

**Lemma 2.** *If perturbation $dK$ is small compared to $K$, then*

$$dL = (1 + o(1)) \Delta D^{-2} K - D^{-1} dK. \tag{7}$$

Then, element-wise, the first two moments of $dL$ can be estimated as

$$
\begin{align}
\mathrm{E}(dL) &\approx \mathrm{E}(\Delta) D^{-2} K - D^{-1} \mathrm{E}(dK) \tag{8} \\
\mathrm{E}(dL^2) &\approx \mathrm{E}\left(\Delta D^{-2} K \circ \Delta D^{-2} K - 2D^{-1} dK \circ \Delta D^{-2} K + D^{-1} dK \circ D^{-1} dK\right) \\
&= \mathrm{E}\left(\Delta^2\right) D^{-4} K^2 + D^{-2} \mathrm{E}\left(dK^2\right) - 2\mathrm{E}(\Delta dK) D^{-3} \circ K, \tag{9}
\end{align}
$$

where $\circ$ denotes element-wise product. The quantities needed to estimate $\mathrm{E}(dL)$ and $\mathrm{E}(dL^2)$ can be obtained from moments and correlations among the elements of the similarity matrix $\tilde{K}_{ij}$. In particular, we have

$$
\mathrm{E}(dK_{ij}) = \mathrm{E}\left(\tilde{K}_{ij}\right) - K_{ij}, \quad \mathrm{E}(dK_{ij})^2 = \mathrm{E}\tilde{K}_{ij}^2 - 2K_{ij} \mathrm{E}\left(\tilde{K}_{ij}\right) + K_{ij}^2 \tag{10}
$$

$$
\mathrm{E}\Delta_i = \mathrm{E}\tilde{D}_i - D_i, \quad \mathrm{E}\tilde{D}_i = \sum_{j=1}^{n} \mathrm{E}\left(\tilde{K}_{ij}\right), \quad \mathrm{E}\Delta_i^2 = \mathrm{E}\tilde{D}_i^2 - 2D_i \cdot \mathrm{E}\tilde{D}_i + D_i^2 \tag{11}
$$

$$
\mathrm{E}\tilde{D}_i^2 = \mathrm{E}\left(\sum_{j=1}^{n} \tilde{K}_{ij}\right)^2 = \sum_{j=1}^{n} \mathrm{E}\tilde{K}_{ij}^2 + 2\sum_{j=1}^{n} \sum_{q=j+1}^{n} \left(\mathrm{E}\tilde{K}_{ij} \mathrm{E}\tilde{K}_{iq} + \rho_{ijq}^k \sigma_{ij}^k \sigma_{iq}^k\right) \tag{12}
$$

$$
\begin{align}
\mathrm{E}(\Delta dK)_{ij} &= \mathrm{E}(\tilde{D}_i - D_i)(\tilde{K}_{ij} - K_{ij}) = \mathrm{E}\left(\tilde{D}_i \tilde{K}_{ij}\right) - D_i \mathrm{E}\tilde{K}_{ij} - K_{ij} \mathrm{E}\Delta_i \\
&= \mathrm{E}\left[\tilde{K}_{ij}^2 + \tilde{K}_{ij}\left(\sum_{q=1,q\neq j}^{n} \tilde{K}_{iq}\right)\right] - D_i \mathrm{E}\tilde{K}_{ij} - K_{ij} \mathrm{E}\Delta_i \\
&= \mathrm{E}\tilde{K}_{ij}^2 + \sum_{q=1,q\neq j}^{n} \left(\mathrm{E}\tilde{K}_{ij} \mathrm{E}\tilde{K}_{iq} + \rho_{ijq}^k \sigma_{ij}^k \sigma_{iq}^k\right) - D_i \mathrm{E}\tilde{K}_{ij} - K_{ij} \mathrm{E}\Delta_i, \tag{13}
\end{align}
$$

where $\sigma_{ij}^k$ is the standard deviation of $\tilde{K}_{ij}$ and $-1 \leq \rho_{ijq}^k \leq 1$ is the correlation coefficient between $\tilde{K}_{ij}$ and $\tilde{K}_{iq}$. Estimating all $\rho_{ijq}^k{}'s$ would require an intensive effort. For simplicity, we could set $\rho_{ijq}^k$ to 1 in Eq. (12) and to $-1$ in Eq. (13), and obtain an upper bound for $\mathrm{E}(dL^2)$. This bound could optionally be tightened by using a simulation method to estimate the values of $\rho_{ijq}^k$. However, in our experimental work we have found that our results are insensitive to the values of $\rho_{ijq}^k$, and setting $\rho_{ijq}^k = 0.5$ usually achieves good results.

**Remark.** Eqs. (8)–(13) allow us to estimate (i.e., to upper bound) the first two moments of $dL$ using those of $dK$, which are computed using Eq. (15) or (16) in Section 3.4.

### 3.4 Perturbation on the similarity matrix

The similarity matrix $\tilde{K}$ on perturbed data $\tilde{X}$ is

$$\tilde{K}_{ij} = \exp\left(-\frac{||\mathbf{x}_i - \mathbf{x}_j + \epsilon_i - \epsilon_j||^2}{2\sigma_k^2}\right), \tag{14}$$

where $\sigma_k$ is the kernel bandwidth. Then, given data $X$, the first two moments of $dK_{ij} = \tilde{K}_{ij} - K_{ij}$, the error in the similarity matrix, can be determined by one of the following lemmas.

**Lemma 3.** *Given $X$, if all components of $\epsilon_i$ and $\epsilon_j$ are i.i.d. Gaussian $N(0, \sigma^2)$, then*

$$\mathrm{E}\left(\tilde{K}_{ij}\right) = M_{ij}\left(-\frac{\sigma^2}{\sigma_k^2}\right), \quad \mathrm{E}\left(\tilde{K}_{ij}^2\right) = M_{ij}\left(-\frac{2\sigma^2}{\sigma_k^2}\right), \tag{15}$$

*where $M_{ij}(t) = \left[\exp\left(\frac{\lambda_{ij} t}{1-2t}\right)/(1 - 2t)^{d/2}\right]$, and $\lambda_{ij} = \left(||\mathbf{x}_i - \mathbf{x}_j||^2/2\sigma^2\right)$.*

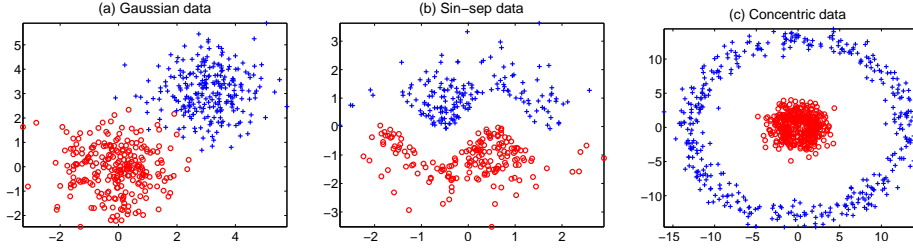

Figure 6: Synthetic data sets illustrated in two dimensions.

**Lemma 4.** *Under Assumption* A*, given* $X$ *and for large values of the dimension d, the first two moments of* $\tilde{K}_{ij}$ *can be computed approximately as follows:*

$$\mathrm{E}\left(\tilde{K}_{ij}\right) = M_{ij}\left(-\frac{1}{2\sigma_k^2}\right), \quad \mathrm{E}\left(\tilde{K}_{ij}^2\right) = M_{ij}\left(-\frac{1}{\sigma_k^2}\right), \tag{16}$$

*where* $M_{ij}(t) = \exp\left[\left(\lambda_{ij} + 2d\sigma^2\right)t + \left(d\mu^4 + d\sigma^4 + 4\sigma^2\lambda_{ij}^2\right)t^2\right]$*, and* $\lambda_{ij} = ||\mathbf{x}_i - \mathbf{x}_j||^2$.

**Remark.** (i) Given data perturbation error $\sigma$, kernel bandwidth $\sigma_k$ and data $X$, the first two moments of $dK_{ij}$ can be estimated directly using (15) or (16). (ii) Through Eqs. (1)–(16), we have established a relationship between the mis-clustering rate $\eta$ and the data perturbation magnitude $\sigma$. By inverting this relationship (e.g., using binary search), we can determine a $\sigma^*$ for a given $\eta^*$.

## 4 Evaluation

In this section we present an empirical evaluation of our analysis on 3 synthetic data sets (see Fig. 6) and 6 real data sets from the UCI repository [14]. The data domains are diverse, including image, medicine, agriculture, etc., and the different data sets impose different difficulty levels on the underlying spectral clustering algorithm, demonstrating the wide applicability of our analysis.

In the experiments, we use data quantization as the perturbation scheme to evaluate the upper bound provided by our analysis on the clustering error. Fig. 7 plots the mis-clustering rate and the upper bound for data sets subject to varying degrees of quantization. As expected, the mis-clustering rate increases as one decreases the number of quantization bits. We find that the error bounds are remarkably tight, which validate the assumptions we make in the analysis. It is also interesting to note that even when using as few as 3-4 bits, the clustering degrades very little in both real error and as assessed by our bound. The effectiveness of our bound should allow the practitioner to determine the right amount of quantization given a permitted loss in clustering performance.

## 5 Conclusion

In this paper, we proposed a theoretical analysis of the clustering error for spectral clustering in the face of stochastic perturbations. Our experimental evaluation has provided support for the assumptions made in the analysis, showing that the bound is tight under conditions of practical interest. We believe that our work, which provides an analytical relationship between the mis-clustering rate and the variance of the perturbation, constitutes a critical step towards enabling a large class of applications that seek to perform clustering of objects, machines, data, etc in a distributed environment. Many networks are bandwidth constrained, and our methods can guide the process of data thinning so as to limit the amount of data transmitted through the network for the purpose of clustering.

## References

[1] L. Bottou and O. Bousquet, "The tradeoffs of large scale learning," in *Advances in Neural Information Processing Systems 20*, 2007.

[2] A. Silberstein, G. P. A. Gelfand, K. Munagala, and J. Yang, "Suppression and failures in sensor networks: A Bayesian approach," in *Proceedings of VLDB*, 2007.

[3] X. Nguyen, M. J. Wainwright, and M. I. Jordan, "Nonparametric decentralized detection using kernel methods," *IEEE Transactions on Signal Processing*, vol. 53, no. 11, pp. 4053–4066, 2005.

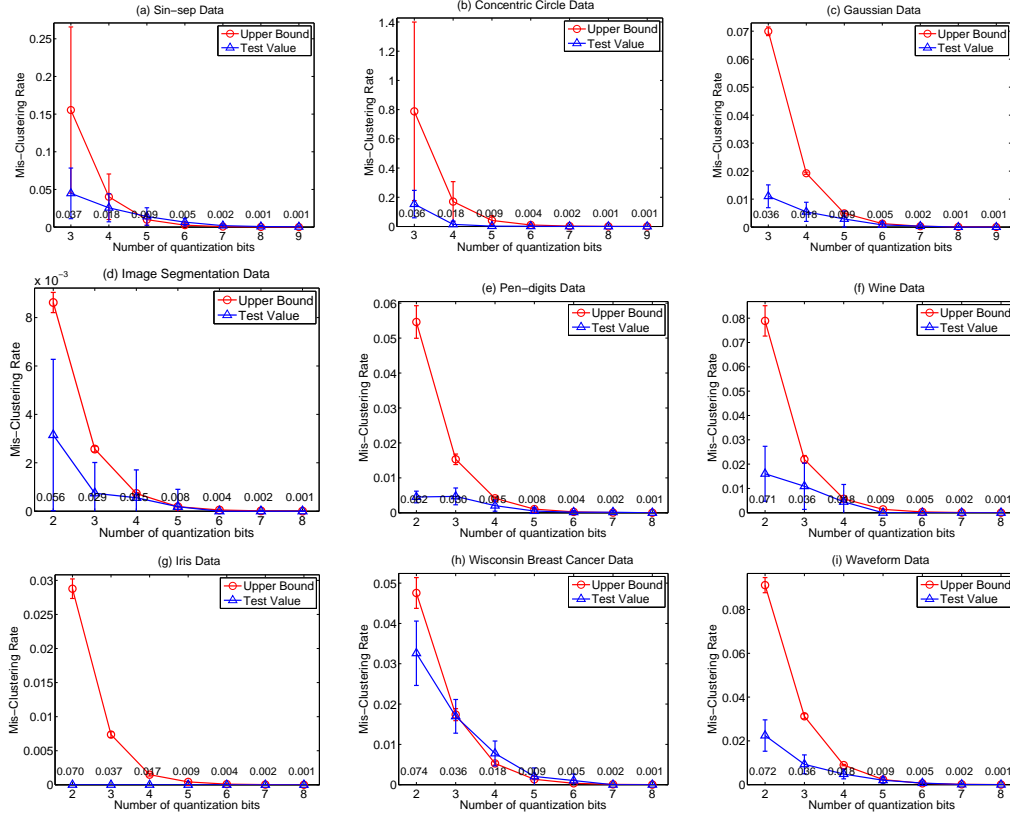

Figure 7: Upper bounds of clustering error on approximate data obtained from quantization as a function of the number of bits. (a–c) Simulated data sets (1000 sample size, 2, 2, 10 features, respectively); (d) Statlog image segmentation data (2310 sample size, 19 features); (e) Handwritten digits data (10992 sample size, 16 features); (f) Wine data (178 sample size, 13 features); (g) Iris data (150 sample size, 4 features). (h) Wisconsin breast cancer data (569 sample size, 30 features); (i) Waveform data (5000 sample size, 21 features). The $x$-axis shows the number of quantization bits and (above the axis) the corresponding data perturbation error $\sigma$. Error bars are derived from 25 replications. In the experiments, all data values are normalized in range $[0, 1]$. For data sets with more than two clusters, we choose two of them for the experiments.

[4] L. Huang, X. Nguyen, M. Garofalakis, A. D. Joseph, M. I. Jordan, and N. Taft, "In-network PCA and anomaly detection," in *Advances in Neural Information Processing Systems (NIPS)*, 2006.

[5] R. Kannan, S. Vempala, and A. Vetta, "On clusterings: Good, bad and spectral," *Journal of the ACM*, vol. 51, no. 3, pp. 497–515, 2004.

[6] A. Y. Ng, M. Jordan, and Y. Weiss, "On spectral clustering: Analysis and an algorithm," in *Advances in Neural Information Processing Systems (NIPS)*, 2002.

[7] J. Shi and J. Malik, "Normalized cuts and image segmentation," *IEEE Transactions on Pattern Analysis and Machine Intelligence*, vol. 22, no. 8, pp. 888–905, 2000.

[8] U. von Luxburg, M. Belkin, and O. Bousquet, "Consistency of spectral clustering," *Annals of Statistics*, vol. 36, no. 2, pp. 555–586, 2008.

[9] P. Drineas and M. W. Mahoney, "On the Nyström method for approximating a Gram matrix for improved kernel-based learning," in *Proceedings of COLT*, 2005, pp. 323–337.

[10] C. Fowlkes, S. Belongie, F. Chung, and J. Malik, "Spectral grouping using the Nyström method," *IEEE Transactions on Pattern Analysis and Machine Intelligence*, vol. 26, no. 2, 2004.

[11] G. Cormode and M. Garofalakis, "Sketching streams through the net: Distributed approximate query tracking," in *Proceedings of VLDB*, 2005, pp. 13–24.

[12] D. Kushnir, M. Galun, and A. Brandt, "Fast multiscale clustering and manifold identification," *Pattern Recognition*, vol. 39, no. 10, pp. 1876–1891, 2006.

[13] G. W. Stewart and J. Guang Sun, *Matrix Perturbation Theory*. Academic Press, 1990.

[14] A. Asuncion and D. Newman, "UCI Machine Learning Repository, Department of Information and Computer Science," 2007, http://www.ics.uci.edu/ mlearn/MLRepository.html.

